# Reconstruction of Sequential Data with Probabilistic Models and Continuity Constraints

**Miguel Á. Carreira-Perpiñán**
Dept. of Computer Science, University of Sheffield, UK
*miguel@dcs.shef.ac.uk*

## Abstract

We consider the problem of reconstructing a temporal discrete sequence of multidimensional real vectors when part of the data is missing, under the assumption that the sequence was generated by a continuous process. A particular case of this problem is multivariate regression, which is very difficult when the underlying mapping is one-to-many. We propose an algorithm based on a joint probability model of the variables of interest, implemented using a nonlinear latent variable model. Each point in the sequence is potentially reconstructed as any of the modes of the conditional distribution of the missing variables given the present variables (computed using an exhaustive mode search in a Gaussian mixture). Mode selection is determined by a dynamic programming search that minimises a geometric measure of the reconstructed sequence, derived from continuity constraints. We illustrate the algorithm with a toy example and apply it to a real-world inverse problem, the acoustic-to-articulatory mapping. The results show that the algorithm outperforms conditional mean imputation and multilayer perceptrons.

## 1 Definition of the problem

Consider a mobile point following a continuous trajectory in a subset of $\mathbb{R}^D$. Imagine that it is possible to obtain a finite number of measurements of the position of the point. Suppose that these measurements are corrupted by noise and that sometimes part of, or all, the variables are missing. The problem considered here is to reconstruct the sequence from the part of it which is observed. In the particular case where the present variables and the missing ones are the same for every point, the problem is one of multivariate regression. If the pattern of missing variables is more general, the problem is one of missing data reconstruction.

Consider the problem of regression. If the present variables uniquely identify the missing ones at every point of the data set, the problem can be adequately solved by a universal function approximator, such as a multilayer perceptron. In a probabilistic framework, the conditional mean of the missing variables given the present ones will minimise the average squared reconstruction error [3]. However, if the underlying mapping is one-to-many, there will be regions in the space for which the present variables do not identify uniquely the missing ones. In this case, the conditional mean mapping will fail, since it will give a compromise value—an average of the correct ones. *Inverse problems*, where the inverse

of a mapping is one-to-many, are of this type. They include the acoustic-to-articulatory mapping in speech [15], where different vocal tract shapes may produce the same acoustic signal, or the robot arm problem [2], where different configurations of the joint angles may place the hand in the same position.

In some situations, data reconstruction is a means to some other objective, such as classification or inference. Here, we deal solely with data reconstruction of temporally continuous sequences according to the squared error. Our algorithm does not apply for data sets that either lack continuity (e.g. discrete variables) or have lost it (e.g. due to undersampling or shuffling).

We follow a statistical learning approach: we attempt to reconstruct the sequence by learning the mapping from a training set drawn from the probability distribution of the data, rather than by solving a physical model of the system. Our algorithm can be described briefly as follows. First, a joint density model of the data is learned in an unsupervised way from a sample of the data[1]. Then, pointwise reconstruction is achieved by computing all the modes of the conditional distribution of the missing variables given the present ones at the current point. In principle, any of these modes is potentially a plausible reconstruction. When reconstructing a sequence, we repeat this mode search for every point in the sequence, and then find the combination of modes that minimises a geometric sequence measure, using dynamic programming. The sequence measure is derived from local continuity constraints, e.g. the curve length.

The algorithm is detailed in §2 to §4. We illustrate it with a 2D toy problem in §5 and apply it to an acoustic-to-articulatory-like problem in §6. §7 discusses the results and compares the approach with previous work.

Our notation is as follows. We represent the observed variables in vector form as $\mathbf{t} = (t_1, \ldots, t_D) \in \mathbb{R}^D$. A data set (possibly a temporal sequence) is represented as $\{\mathbf{t}_n\}_{n=1}^N$. Groups of variables are represented by sets of indices $\mathcal{I}, \mathcal{J} \in \{1, \ldots, D\}$, so that if $\mathcal{I} = \{1, 7, 3\}$, then $\mathbf{t}_\mathcal{I} = (t_1 t_7 t_3)$.

## 2  Joint generative modelling using latent variables

Our starting point is a joint probability model of the observed variables $p(\mathbf{t})$. From it, we can compute conditional distributions of the form $p(\mathbf{t}_\mathcal{J}|\mathbf{t}_\mathcal{I})$ and, by picking representative points, derive a (multivalued) mapping $\mathbf{t}_\mathcal{I} \to \mathbf{t}_\mathcal{J}$. Thus, contrarily to other approaches, e.g. [6], we adopt *multiple pointwise imputation*. In §4 we show how to obtain a single reconstructed sequence of points.

Although density estimation requires more parameters than mapping approximation, it has a fundamental advantage [6]: the density model represents the relation between any variables, which allows to choose any missing/present variable combination. A mapping approximator treats asymmetrically some variables as inputs (present) and the rest as outputs (missing) and can't easily deal with other relations.

The existence of functional relationships (even one-to-many) between the observed variables indicates that the data must span a low-dimensional manifold in the data space. This suggests the use of latent variable models for modelling the joint density. However, it is possible to use other kinds of density models.

In latent variable modelling the assumption is that the observed high-dimensional data $\mathbf{t}$ is generated from an underlying low-dimensional process defined by a small number $L$ of *latent variables* $\mathbf{x} = (x_1, \ldots, x_L)$ [1]. The latent variables are mapped by a fixed

transformation into a $D$-dimensional data space and noise is added there. A particular model is specified by three parametric elements: a prior distribution in latent space $p(\mathbf{x})$, a smooth mapping $\mathbf{f}$ from latent space to data space and a noise model in data space $p(\mathbf{t}|\mathbf{x})$. Marginalising the joint probability density function $p(\mathbf{t}, \mathbf{x})$ over the latent space gives the distribution in data space, $p(\mathbf{t})$. Given an observed sample in data space $\{\mathbf{t}_n\}_{n=1}^{N}$, a parameter estimate can be found by maximising the log-likelihood, typically using an EM algorithm. We consider the following latent variable models, both of which allow easy computation of conditional distributions of the form $p(\mathbf{t}_{\mathcal{J}}|\mathbf{t}_{\mathcal{I}})$:

**Factor analysis** [1], in which the mapping is linear, the prior in latent space is unit Gaussian and the noise model is diagonal Gaussian. The density in data space is then Gaussian with a constrained covariance matrix. We use it as a baseline for comparison with more sophisticated models.

**The generative topographic mapping (GTM)** [4] is a nonlinear latent variable model, where the mapping is a generalised linear model, the prior in latent space is discrete uniform and the noise model is isotropic Gaussian. The density in data space is then a constrained mixture of isotropic Gaussians.

In latent variable models that sample the latent space prior distribution (like GTM), the mixture centroids in data space (associated to the latent space samples) are not trainable parameters. We can then improve the density model at a higher computational cost with no generalisation loss by increasing the number of mixture components. Note that the number of components required will depend exponentially on the intrinsic dimensionality of the data (ideally coincident with that of the latent space, $L$) and not on the observed one, $D$.

## 3   Exhaustive mode finding

Given a conditional distribution $p(\mathbf{t}_{\mathcal{J}}|\mathbf{t}_{\mathcal{I}})$, we consider all its modes as plausible predictions for $\mathbf{t}_{\mathcal{J}}$. This requires an exhaustive mode search in the space of $\mathbf{t}_{\mathcal{J}}$. For Gaussian mixtures, we do this by using a maximisation algorithm starting from each centroid[2], such as a fixed-point iteration or gradient ascent combined with quadratic optimisation [5]. In the particular case where all variables are missing, rather than performing a mode search, we return as predictions all the component centroids. It is also possible to obtain error bars at each mode by locally approximating the density function by a normal distribution. However, if the dimensionality of $\mathbf{t}_{\mathcal{J}}$ is high, the error bars become very wide due to the curse of the dimensionality.

An advantage of multiple pointwise imputation is the easy incorporation of extra constraints on the missing variables. Such constraints might include keeping only those modes that lie in an interval dependent on the present variables [8] or discarding low-probability (spurious) modes—which speeds up the reconstruction algorithm and may make it more robust.

A faster way to generate representative points of $p(\mathbf{t}_{\mathcal{J}}|\mathbf{t}_{\mathcal{I}})$ is simply to draw a fixed number of samples from it—which may also give robustness to poor density models. However, in practice this resulted in a higher reconstruction error.

## 4   Continuity constraints and dynamic programming (DP) search

Application of the exhaustive mode search to the conditional distribution at every point of the sequence produces one or more candidate reconstructions per point. To select a

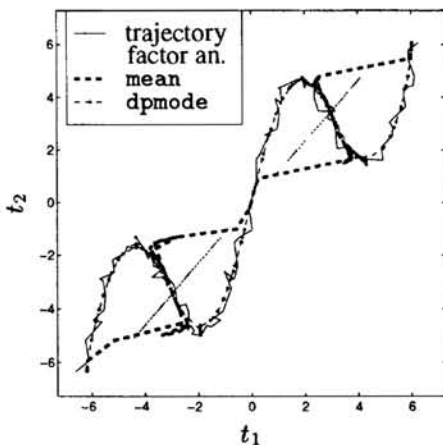

Average squared reconstruction error

| Missing pattern | Factor analysis | MLP[a] | GTM | | |
|---|---|---|---|---|---|
| | | | mean | dpmode | cmode |
| $t_2$ | 3.8902 | 0.2046 | 0.2044 | 0.2168 | 0.2168 |
| $t_1$ | 4.3226 | 2.5126 | 2.4224 | 0.0522 | 0.0522 |
| $t_1$ or $t_2$ | 4.2020 | — | 1.2963 | 0.1305 | 0.1305 |
| 10% | 1.0983 | — | 0.3970 | 0.0253 | 0.0251 |
| 50% | 6.2914 | — | 4.6530 | 0.1176 | 0.0771 |
| 90% | 21.4942 | — | 20.7877 | 2.2261 | 0.0643 |

[a]The MLP cannot be applied to varying patterns of missing data.

Table 1: Trajectory reconstruction for a 2D problem. The table gives the average squared reconstruction error when $t_2$ is missing (row 1), $t_1$ is missing (row 2), exactly one variable per point is missing at random (row 3) or a percentage of the values are missing at random (rows 4–6). The graph shows the reconstructed trajectory when $t_1$ is missing: factor analysis (straight, dotted line), mean (thick, dashed), dpmode (superimposed on the trajectory).

single reconstructed sequence, we define a *local continuity constraint*: consecutive points in time should also lie nearby in data space. That is, if $\delta$ is some suitable distance in $\mathbb{R}^D$, $\delta(\mathbf{t}_n, \mathbf{t}_{n+1})$ should be small. Then we define a *global geometric measure* $\mathscr{C}$ for a sequence $\{\mathbf{t}_n\}_{n=1}^{N}$ as $\mathscr{C}\left(\{\mathbf{t}_n\}_{n=1}^{N}\right) \stackrel{\text{def}}{=} \sum_{n=1}^{N-1} \delta(\mathbf{t}_n, \mathbf{t}_{n+1})$. We take $\delta$ as the Euclidean distance, so $\mathscr{C}$ becomes simply the length of the sequence (considered as a polygonal line). Finding the sequence of modes with minimal $\mathscr{C}$ is efficiently achieved by dynamic programming.

## 5   Results with a toy problem

To illustrate the algorithm, we generated a 2D data set from the curve $(t_1, t_2) = (x, x + 3\sin(x))$ for $x \in [-2\pi, 2\pi]$, with normal isotropic noise (standard deviation 0.2) added. Thus, the mapping $t_1 \to t_2$ is one-to-one but the inverse one, $t_2 \to t_1$, is multivalued. One-dimensional factor analysis (6 parameters) and GTM models (21 parameters) were estimated from a 1000-point sample, as well as two 48-hidden-unit multilayer perceptrons (98 parameters), one for each mapping. For GTM we tried several strategies to select points from the conditional distribution: mean (the conditional mean), dpmode (the mode selected by dynamic programming) and cmode (the closest mode to the actual value of the missing variable). The cmode, unknown in practice, is used here to compute a lower bound on the performance of any mode-based strategy. Other strategies, such as picking the global mode, a random mode or using a local (greedy) search instead of dynamic programming, gave worse results than the dpmode.

Table 1 shows the results for reconstructing a 100-point trajectory. The nonlinear nature of the problem causes factor analysis to break down in all cases. For the one-to-one mapping case ($t_2$ missing) all the other methods perform well and recover the original trajectory, with mean attaining the lowest error, as predicted by the theory[3]. For the one-to-many case ($t_1$ missing, see fig.), both the MLP and the mean are unable to track more than one branch of the mapping, but the dpmode still recovers the original mapping. For random missing

| Missing | Factor | GTM | | |
|---------|--------|------|--------|-------|
| pattern | analysis | mean | dpmode | cmode |
| PLP | 0.9165 | 0.6217 | 0.6250 | 0.4587 |
| EPG | 3.7177 | 2.3729 | 2.0613 | 1.0538 |
| 10% | 0.2046 | 0.0947 | 0.0903 | 0.0841 |
| 50% | 1.1285 | 0.7540 | 0.6527 | 0.6023 |
| blocks | 0.1950 | 0.1669 | 0.1005 | 0.0925 |

Table 2: Average squared reconstruction error for an utterance. The last row corresponds to a missing pattern of square blocks totalling 10% of the utterance.

patterns[4], the dpmode is able to cope well with high amounts of missing data.

The consistently low error of the cmode shows that the modes contain important information about the possible options to predict the missing values. The performance of the dpmode, close to that of the cmode even for large amounts of missing data, shows that application of the continuity constraint allows to recover that information.

## 6   Results with real speech data

We report a preliminary experiment using acoustic and electropalatographic (EPG) data[5] for the utterance "Put your hat on the hatrack and your coat in the cupboard" (speaker FG) from the ACCOR database [10]. 12th-order perceptual linear prediction coefficients [7] plus the log-energy were computed at 200 Hz from its acoustic waveform. The EPG data consisted of 62-bit frames sampled at 200 Hz, which we consider as 62-dimensional vectors of real numbers. No further preprocessing of the data was carried out. Thus, the resulting sequence consisted of over 600 75-dimensional real vectors. We constructed a training set by picking, in random order, 80% of these vectors. The whole utterance was used for the reconstruction test.

We trained two density models: a 9-dimensional factor analysis (825 parameters) and a two-dimensional[6] GTM (3676 parameters) with a $20 \times 20$ grid (resulting in a mixture of 400 isotropic Gaussians in the 75-dimensional data space). Table 2 confirms again that the linear method (factor analysis) fares worst (despite its use of a latent space of dimension $L = 9$). The dpmode attains almost always a lower error than the conditional mean, with up to a 40% improvement (the larger the higher the amount of missing data). When a shuffled version of the utterance (thus having lost its continuity) was reconstructed, the error of the dpmode was consistently higher than that of the mean, indicating that the application of the continuity constraint was responsible for the error decrease.

## 7   Discussion

Using a joint probability model allows flexible construction of predictive distributions for the missing data: varying patterns of missing data and multiple pointwise imputations are possible, as opposed to standard function approximators. We have shown that the modes of the conditional distribution of the missing variables given the present ones are potentially

plausible reconstructions of the missing values, and that the application of local continuity constraints—when they hold—can help to recover the actually plausible ones.

**Previous work**  The key aspects of our approach are the use of a joint density model (learnt in an unsupervised way), the exhaustive mode search, the definition of a geometric trajectory measure derived from continuity constraints and its implementation by dynamic programming. Several of these ideas have been applied earlier in the literature, which we review briefly.

The use of the joint density model for prediction is the basis of the statistical technique of *multiple imputation* [9]. Here, several versions of the complete data set are generated from the appropriate conditional distributions, analysed by standard complete-data methods and the results combined to produce inferences that incorporate missing-data uncertainty. Ghahramani and Jordan [6] also proposed the use of the joint density model to generate a single estimate of the missing variables and applied it to a classification problem.

Conditional distributions have been approximated by MLPs rather than by density estimation [16], but this lacks flexibility to varying patterns of missing data and requires an extra model of the input variables distribution (unless assumed uniform).

Rohwer and van der Rest [12] introduce a cost function with a description length interpretation whose minimum is approximated by the densest mode of a distribution. A neural network trained with this cost function can learn one branch of a multivariate mapping, but is unable to select other branches which may be correct at a given time.

Continuity constraints implemented via dynamic programming have been used for the acoustic-to-articulatory mapping problem [15]. Reasonable results (better than using an MLP to approximate the mapping) can be obtained using a large codebook of acoustic and articulatory vectors. Rahim et al. [11] achieve similar quality with much less computational requirements using an assembly of MLPs, each one trained in a different area of the acoustic-articulatory space, to locally approximate the mapping. However, clustering the space is heuristic (with no guarantee that the mapping is one-to-one in each region) and training the assembly is difficult. It also lacks flexibility to varying missingness patterns.

A number of trajectory measures have been used in the robot arm problem literature [2] and minimised by dynamic programming, such as the energy, torque, acceleration, jerk, etc.

**Temporal modelling**  It is important to remark that our approach does not attempt to model the temporal evolution of the system. The joint probability model is estimated statically. The temporal aspect of the data appears indirectly and a posteriori through the application of the continuity constraints to select a trajectory[7]. In this respect, our approach differs from that of dynamical systems or from models based in Markovian assumptions, such as hidden Markov models or other trajectory models [13, 14]. However, the fact that the duration or speed of the trajectory plays no role in the algorithm may make it invariant to time warping (e.g. robust to fast/slow speech styles).

**Choice of density model**  The fact that the modes are a key aspect of our approach make it sensitive to the density model. With finite mixtures, spurious modes can appear as ripple superimposed on the density function in regions where the mixture components are sparsely distributed and have little interaction. Such modes can lead the DP search to a wrong trajectory. Possible solutions are to improve the density model (perhaps by increasing the number of components, see §2, or by regularisation), to smooth the conditional distribution or to look for *bumps* (regions of high probability mass) instead of modes.

**Computational cost** The DP search has complexity $\mathcal{O}(NM^2)$, where $M$ is an average of the number of modes per sequence point and $N$ the number of points in the sequence. In our experiments $M$ is usually small and the DP search is fast even for long sequences. The bottleneck of the reconstruction part of the algorithm is obtaining the modes of the conditional distribution for every point in the sequence when there are many missing variables.

**Further work** We envisage more thorough experiments using data from the Wisconsin X-ray microbeam database and comparing with recurrent MLPs or an MLP committee, which may be more suitable for multivalued mappings. Extensions of our algorithm include different geometric measures (e.g. curvature-based rather than length-based), different strategies for multiple pointwise imputation (e.g. bump searching) or multidimensional constraints (e.g. temporal and spatial). Other practical applications include audiovisual mappings for speech, hippocampal place cell reconstruction and wind vector retrieval from scatterometer data.

## Acknowledgments

We thank Steve Renals for useful conversations and for comments about this paper.

## Footnotes

[1]In our examples we only use complete training data (i.e., with no missing data), but it is perfectly possible to estimate a probability model with incomplete training data by using an EM algorithm [6].

[2]Actually, given a value of $\mathbf{t}_{\mathcal{I}}$, most centroids have negligible posterior probability and can be removed from the mixture with practically no loss of accuracy. Thus, a large number of mixture components may be used without deteriorating excessively the computational efficiency.

[3]A combined strategy could retain the optimality of the mean in the one-to-one case and the advantage of the modes in the one-to-many case, by choosing the conditional mean (rather than the mode) when the conditional distribution is unimodal, and all the modes otherwise.

[4]Note that the nature of the missing pattern (missing at random, missing completely at random, etc. [9]) does not matter for reconstruction—although it does for estimation.

[5]An EPG datum is the (binary) contact pattern between the tongue and the palate at selected locations in the latter. Note that it is an incomplete articulatory representation of speech.

[6]A latent space of 2 dimensions is clearly too low for this data, but the computational complexity of GTM prevents the use of a higher one. Still, its nonlinear character compensates partly for this.

[7]However, the method may be derived by assuming a distribution over the whole sequence with a normal, Markovian dependence between adjacent frames.

## References

[1] D. J. Bartholomew. *Latent Variable Models and Factor Analysis*. Charles Griffin & Company Ltd., London, 1987.

[2] N. Bernstein. *The Coordination and Regulation of Movements*. Pergamon, Oxford, 1967.

[3] C. M. Bishop. *Neural Networks for Pattern Recognition*. Oxford University Press, 1995.

[4] C. M. Bishop, M. Svensén, and C. K. I. Williams. GTM: The generative topographic mapping. *Neural Computation*, 10(1):215–234, Jan. 1998.

[5] M. Á. Carreira-Perpiñán. Mode-finding in Gaussian mixtures. Technical Report CS–99–03, Dept. of Computer Science, University of Sheffield, UK, Mar. 1999. Available online at http://www.dcs.shef.ac.uk/~miguel/papers/cs-99-03.html.

[6] Z. Ghahramani and M. I. Jordan. Supervised learning from incomplete data via an EM approach. In *NIPS 6*, pages 120–127, 1994.

[7] H. Hermansky. Perceptual linear predictive (PLP) analysis of speech. *J. Acoustic Soc. Amer.*, 87(4):1738–1752, Apr. 1990.

[8] L. Josifovski, M. Cooke, P. Green, and A. Vizinho. State based imputation of missing data for robust speech recognition and speech enhancement. In *Proc. Eurospeech 99*, pages 2837–2840, 1999.

[9] R. J. A. Little and D. B. Rubin. *Statistical Analysis with Missing Data*. John Wiley & Sons, New York, London, Sydney, 1987.

[10] A. Marchal and W. J. Hardcastle. ACCOR: Instrumentation and database for the cross-language study of coarticulation. *Language and Speech*, 36(2, 3):137–153, 1993.

[11] M. G. Rahim, C. C. Goodyear, W. B. Kleijn, J. Schroeter, and M. M. Sondhi. On the use of neural networks in articulatory speech synthesis. *J. Acoustic Soc. Amer.*, 93(2):1109–1121, Feb. 1993.

[12] R. Rohwer and J. C. van der Rest. Minimum description length, regularization, and multimodal data. *Neural Computation*, 8(3):595–609, Apr. 1996.

[13] S. Roweis. Constrained hidden Markov models. In *NIPS 12 (this volume)*, 2000.

[14] L. K. Saul and M. G. Rahim. Markov processes on curves for automatic speech recognition. In *NIPS 11*, pages 751–757, 1999.

[15] J. Schroeter and M. M. Sondhi. Techniques for estimating vocal-tract shapes from the speech signal. *IEEE Trans. Speech and Audio Process.*, 2(1):133–150, Jan. 1994.

[16] V. Tresp, R. Neuneier, and S. Ahmad. Efficient methods for dealing with missing data in supervised learning. In *NIPS 7*, pages 689–696, 1995.